# Handwritten Word Recognition using Contextual Hybrid Radial Basis Function Network/Hidden Markov Models

**Bernard Lemarié**
La Poste/SRTP
10, Rue de l'île-Mabon
F-44063 Nantes Cedex France
lemarie@srtp.srt-poste.fr

**Michel Gilloux**
La Poste/SRTP
10, Rue de l'île-Mabon
F-44063 Nantes Cedex France
gilloux@srtp.srt-poste.fr

**Manuel Leroux**
La Poste/SRTP
10, Rue de l'île-Mabon
F-44063 Nantes Cedex France
leroux@srtp.srt-poste.fr

## Abstract

A hybrid and contextual radial basis function network/hidden Markov model off-line handwritten word recognition system is presented. The task assigned to the radial basis function networks is the estimation of emission probabilities associated to Markov states. The model is contextual because the estimation of emission probabilities takes into account the left context of the current image segment as represented by its predecessor in the sequence. The new system does not outperform the previous system without context but acts differently.

## 1 INTRODUCTION

Hidden Markov models (HMMs) are now commonly used in off-line recognition of handwritten words (Chen et al., 1994) (Gilloux et al., 1993) (Gilloux et al. 1995a). In some of these approaches (Gilloux et al. 1993), word images are transformed into sequences of image segments through some explicit segmentation procedure. These segments are passed on to a module which is in charge of estimating the probability for each segment to appear when the corresponding hidden state is some state s (state emission probabilities). Model probabilities are generally optimized for the Maximum Likelihood Estimation (MLE) criterion.

MLE training is known to be sub-optimal with respect to discrimination ability when the underlying model is not the true model for the data. Moreover, estimating the emission probabilities in regions where examples are sparse is difficult and estimations may not be accurate. To reduce the risk of over training, images segments consisting of bitmaps are often replaced by feature vector of reasonable length (Chen et al., 1994) or even discrete symbols (Gilloux et al., 1993).

In a previous paper (Gilloux et al., 1995b) we described a hybrid HMM/radial basis function system in which emission probabilities are computed from full-fledged bitmaps though the use of a radial basis function (RBF) neural network. This system demonstrated better recognition rates than a previous one based on symbolic features (Gilloux et al., 1995b). Yet, many misclassification examples showed that some of the simplifying assumptions made in HMMs were responsible for a significant part of these errors. In particular, we observed that considering each segment independently from its neighbours would hurt the accuracy of the model. For example, figure 1 shows examples of letter *a* when it is segmented in two parts. The two parts are obviously correlated.

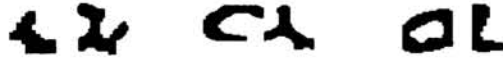

Figure 1: Examples of segmented *a*.

We propose a new variant of the hybrid HMM/RBF model in which emission probabilities are estimated by taking into account the context of the current segment. The context will be represented by the preceding image segment in the sequence.

The RBF model was chosen because it was proven to be an efficient model for recognizing isolated digits or letters (Poggio & Girosi, 1990) (Lemarié, 1993). Interestingly enough, RBFs bear close relationships with gaussian mixtures often used to model emission probabilities in markovian contexts. Their advantage lies in the fact that they do not directly estimate emission probabilities and thus are less prone to errors in this estimation in sparse regions. They are also trained through the Mean Square Error (MSE) criterion which makes them more discriminant.

The idea of using a neural net and in particular a RBF in conjunction with a HMM is not new. In (Singer & Lippman, 1992) it was applied to a speech recognition task. The use of context to improve emission probabilities was proposed in (Bourlard & Morgan, 1993) with the use of a discrete set of context events. Several neural networks are there used to estimate various relations between states, context events and current segment. Our point is to propose a different method without discrete context based on a adapted decomposition of the HMM likelihood estimation.This model is next applied to off-line handwritten word recognition.

The organization of this paper is as follows. Section 1 is an overview of the architecture of our HMM. Section 2 describes the justification for using RBF outputs in a contextual hidden Markov model. Section 3 describes the radial basis function network recognizer. Section 4 reports on an experiment in which the contextual model is applied to the recognition of handwritten words found on french bank or postal cheques.

## 2 OVERVIEW OF THE HIDDEN MARKOV MODEL

In an HMM model (Bahl et al., 1983), the recognition scores associated to words $w$ are likelihoods

$$L(w|i_1...i_n) = p\left(i_1...i_n|w\right) \times p(w)$$

in which the first term in the product encodes the probability with which the model of each word $w$ generates some image (some sequence of image segments) $i_1...i_n$. In the HMM paradigm, this term is decomposed into a sum on all paths (i.e. sequence of hidden states) of products of the probability of the hidden path by the probability that the path generated the image sequence:

$$p(i_1...i_n|w) = \sum_{path=\{s_1...s_n\}} p(i_1...i_n|s_1...s_n) \times p(s_1...s_n)$$

It is often assumed that only one path contributes significantly to this term so that

$$p(i_1 \dots i_n \mid w) = p(i_1 \dots i_n \mid s_1 \dots s_n) \times p(s_1 \dots s_n)$$

In HMMs, each sequence element is assumed to depend only on its corresponding state:

$$p(i_1 \dots i_n \mid s_1 \dots s_n) = \prod_{j=1}^{n} p(i_j \mid s_j)$$

Moreover, first-order Markov models assume that paths are generated by a first-order Markov chain so that

$$p(s_1 \dots s_n) = \prod_{j=1}^{n} p(s_j \mid s_{j-1})$$

We have reported in previous papers (Gilloux et al., 1993) (Gilloux et al., 1995a) on several handwriting recognition systems based on this assumption. The hidden Markov model architecture used in all systems has been extensively presented in (Gilloux et al., 1995a). In that model, word letters are associated to three-states models which are designed to account for the situations where a letter is realized as 0, 1 or 2 segments. Word models are the result of assembling the corresponding letter models. This architecture is depicted on figure 2. We used here transition emission rather than state emission. However, this does not

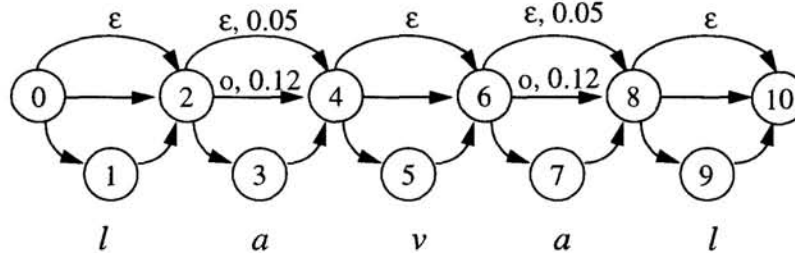

Figure 2: Outline of the model for "*laval*".

change the previous formulas if we replace states by transitions, i.e. pairs of states.

One of these systems was an hybrid RBF/HMM model in which a radial basis function network was used to estimate emission probabilities $p(i_j \mid s_j)$. The RBF outputs are introduced by applying Bayes rule in the expression of $p(i_1 \dots i_n \mid s_1 \dots s_n)$ :

$$p(i_1 \dots i_n \mid s_1 \dots s_n) = \prod_{j=1}^{n} \frac{p(s_j \mid i_j) \times p(i_j)}{p(s_j)}$$

Since the product of a priori image segments probabilities $p(i_j)$ does not depend on the word hypothesis $w$, we may write:

$$p(i_1 \dots i_n \mid s_1 \dots s_n) \propto \prod_{j=1}^{n} \frac{p(s_j \mid i_j)}{p(s_j)}$$

In the above formula, terms of form $p(s_j \mid s_{j-1})$ are transition probabilities which may be estimated through the Baum-Welch re-estimation algorithm. Terms of form $p(s_j)$ are a priori probabilities of states. Note that for Bayes rule to apply, these probabilities have and only have to be estimated consistently with terms of form $p(s_j \mid i_j)$ since $p(i_j \mid s_j)$ is independent of the statistical distribution of states.

It has been proven elsewhere (Richard & Lippman, 1992) that systems trained through the MSE criterion tend to approximate Bayes probabilities in the sense that Bayes proba-

bilities are optimal for the MSE criterion. In practice, the way in which a given system comes close to Bayes optimum is not easily predictable due to various biases of the trained system (initial parameters, local optimum, architecture of the net, etc.). Thus real output scores are generally not equal to Bayes probabilities. However, there exist different procedures which act as a post-processor for outputs of a system trained with the MSE and make them closer to Bayes probabilities (Singer & Lippman, 1992). Provided that such a post-processor is used, we will assume that terms $p(s_j \mid i_j)$ are well estimated by the post-processed outputs of the recognition system. Then, terms $p(s_j)$ are just the a priori probabilities of states on the set used to train the system or post-process the system outputs.

This hybrid handwritten word recognition system demonstrated better performances than previous systems in which word images were represented through sequences of symbolic features instead of full-fledged bitmaps (Gilloux et al., 1995b). However, some recognition errors remained, many of which could be explained by the simplifying assumptions made in the model. In particular, the fact that emission probabilities depend only on the state corresponding to the current bitmap appeared to be a poor choice. For example, on figure 3 the third and fourth segment are classified as two halves of the letter $i$. For letters

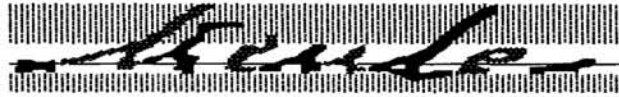

Figure 3: An image of *trente* classified as *mille*.

segmented in two parts, the second half is naturally correlated to the first (see figure 1). Yet, our Markov model architecture is designed so that both halves are assumed uncorrelated. This has two effects. Two consecutive bitmaps which cannot be the two parts of a unique letter are sometimes recognized as such like on figure 3. Also, the emission probability of the second part of a segmented letter is lower than if the first part has been considered for estimating this probability. The contextual model described in the next section is designed so has to make a different assumption on emission probabilities.

## 3 THE HYBRID CONTEXTUAL RBF/HMM MODEL

The exact decomposition of the emission part of word likelihoods is the following:

$$p(i_1 \ldots i_n \mid s_1 \ldots s_n) = p(i_1 \mid s_1 \ldots s_n) \times \prod_{j=2}^{n} p(i_j \mid s_1 \ldots s_n, i_1 \ldots i_{j-1})$$

We assume now that bitmaps are conditioned by their state and the previous image in the sequence:

$$p(i_1 \ldots i_n \mid s_1 \ldots s_n) \cong p(i_1 \mid s_1) \times \prod_{j=2}^{n} p(i_j \mid s_j, i_{j-1})$$

The RBF is again introduced by applying Bayes rule in the following way:

$$p(i_1 \ldots i_n \mid s_1 \ldots s_n) \cong \frac{p(s_1 \mid i_1) \times p(i_1)}{p(s_1)} \times \prod_{j=2}^{n} \frac{p(s_j \mid i_j, i_{j-1}) \times p(i_j \mid i_{j-1})}{p(s_j \mid i_{j-1})}$$

Since terms of form $p(i_j \mid i_{j-1})$ do not contribute to the discrimination of word hypotheses, we may write:

$$p(i_1 \ldots i_n \mid s_1 \ldots s_n) \propto \frac{p(s_1 \mid i_1)}{p(s_1)} \times \prod_{j=2}^{n} \frac{p(s_j \mid i_j, i_{j-1})}{p(s_j \mid i_{j-1})}$$

The RBF has now to estimate not only terms of form $p(s_j \mid i_j, i_{j-1})$ but also terms like $p(s_j \mid i_{j-1})$ which are no longer computed by mere counting. Two radial basis function networks are then used to estimate these probabilities. Their common architecture is described in the next section.

## 4  THE RADIAL BASIS FUNCTION MODEL

The radial basis function model has been described in (Lemarié, 1993). RBF networks are inspired from the theory of regularization (Poggio & Girosi, 1990). This theory studies how multivariate real functions known on a finite set of points may be approximated at these points in a family of parametric functions under some bias of *regularity*. It has been shown that when this bias tends to select smooth functions in the sense that some linear combination of their derivatives is minimum, there exist an analytical solution which is a linear combination of gaussians centred on the points where the function is known (Poggio & Girosi, 1990). It is straightforward to transpose this paradigm to the problem of learning probability distributions given a set of examples.

In practice, the theory is not tractable since it requires one gaussian per example in the training set. Empirical methods (Lemarié, 1993) have been developed which reduce the number of gaussian centres. Since the theory is no longer applicable when the number of centres is reduced, the parameters of the model (centres and covariance matrices for gaussians, weights for the linear combination) have to be trained by another method, in that case the gradient descent method and the MSE criterion. Finally, the resulting RBF model may be looked at like a particular neural network with three layers. The first is the input layer. The second layer is completely connected to the input layer through connections with unit weights. The transfer functions of cells in the second layer are gaussians applied to the weighed distance between the corresponding centres and the weighed input to the cell. The weight of the distance are analogous to the parameters of a diagonal covariance matrix. Finally, the last layer is completely connected to the second one through weighted connections. Cells in this layer just output the sum of their input.

In our experiments, inputs to the RBF are feature vectors of length 138 computed from the bitmaps of a word segment (Lemarié, 1993). The RBF that estimates terms of form $p(s_j \mid i_j, i_{j-1})$ uses to such vectors as input whereas the second RBF (terms $p(s_j \mid i_{j-1})$) is only fed with the vector associated to $i_{j-1}$. These vectors are inspired from "characteristic loci" methods (Gluksman, 1967) and encode the proportion of white pixels from which a bitmap border can be reached without meeting any black pixel in various of directions.

## 5  EXPERIMENTS

The model has been assessed by applying it to the recognition of words appearing in legal amounts of french postal or bank cheques. The size of the vocabulary is 30 and its perplexity is only 14.3 (Bahl et al., 1983). The training and test bases are made of images of amount words written by unknown writers on real cheques. We used 7 191 images during training and 2 879 different images for test. The image resolution was 300 dpi. The amounts were manually segmented into words and an automatic procedure was used to separate the words from the preprinted lines of the cheque form.

The training was conducted by using the results of the former hybrid system. The segmentation module was kept unchanged. There are 48 140 segments in the training set and 19 577 in the test set. We assumed that the base system is almost always correct when aligning segments onto letter models. We thus used this alignment to label all the segments in the training set and took these labels as the desired outputs for the RBF. We used a set of 63 different labels since 21 letters appear in the amount vocabulary and 3 types of segments are possible for each letter. The outputs of the RBF are directly interpreted as Bayes prob-

abilities without further post-processing.

First of all, we assessed the quality of the system by evaluating its ability to recognize the class of a segment through the value of $p(s_j \mid i_j, i_{j-1})$ and compared it with that of the previous hybrid system. The results of this experiment are reported on table 1 for the test set. They demonstrate the importance of the context and thus its potential interest for a

Table 1: Recognition and confusion rates for segment classifiers

|                             | Recognition rate | Confusion rate | Mean square error |
|-----------------------------|:----------------:|:--------------:|:-----------------:|
| RBF system without context  | 32.6%            | 67.4%          | 0.828             |
| RBF system with context     | 41.7%            | 58.3%          | 0.739             |

word recognition system.

We next compare the performance on word recognition on the data base of 2878 images of words. Results are shown in table 2. The first remark is that the system without context

Table 2: Recognition and confusion rates for the word recognition systems

|                             | Recognition rate | Confusion rate | # Confusions |
|-----------------------------|:----------------:|:--------------:|:------------:|
| RBF system without context  | 81,3%            | 16,7%          | 536          |
| RBF system with context     | 76,3%            | 23,7%          | 683          |

present better results than the contextual system. Some of the difference between the systems with and without context are shown below in figures 4 and 5 and may explain why the contextual system remains at a lower level of performance. The word "huit" and "deux" of figure 4 are well recognized by the system without context but badly identified by the contextual system respectively as "trente" and "franc". The image of the word "huit", for example, is segmented into eight segments and each of the four letters of the word is thus necessarily considered as separated in two parts. The fifth and sixth segments are thus recognized as two halves of the letter "i" by the standard system while the contextual system avoids this decomposition of the letter "i". On the next image, the contextual system proposes "ra" for the second and third segments mainly because of the absence of information on the relative position of these segments. On the other hand, figure 5 shows examples where the contextual system outperforms the system without context. In the first case the latter proposed the class "trois" with two halves on the letter "i" on the fifth and sixth segments. In the second case the context is clearly useful for the recognition on the first letter of the word. Forthcoming experiments will try to combine the two systems so as to benefit of their respective characteritics.

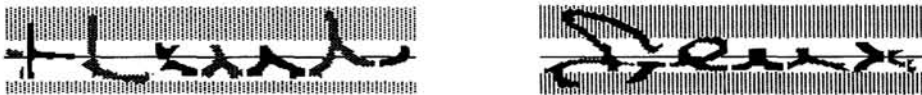

Figure 4 : some new confusions produced by the contextual system.

Experiments have also revealed that the contextual system remains very sensible to the numerical output values for the network which estimates $p(s_j \mid i_{j-1})$. Several approaches for solving this problem are currently under investigation. First results have yet been obtained by trying to approximate the network which estimates $p(s_j \mid i_{j-1})$ from the network which estimates $p(s_j \mid i_j, i_{j-1})$.

## 6 CONCLUSION

We have described a new application of a hybrid radial basis function/hidden Markov model architecture to the recognition of off-line handwritten words. In this architecture, the estimation of emission probabilities is assigned to a discriminant classifier. The estimation of emission probabilities is enhanced by taking into account the context as represented by the previous bitmap in the sequence to be classified. A formula have been derived introducing this context in the estimation of the likelihood of word scores. The ratio of the output values of two networks are now used so as to estimate the likelihood.

The reported experiments reveal that the use of context, if profitable at the segment recognition level, is not yet useful at the word recognition level. Nevertheless, the new system acts differently from the previous system without context and future applications will try to exploit this difference. The dynamic of the ratio networks output values is also very unstable and some solutions to stabilize it which will be deeply tested in the forthcoming experiences.

**References**

Bahl L, Jelinek F, Mercer R, (1983). A maximum likelihood approach to speech recognition. *IEEE Transactions on Pattern Analysis and Machine Intelligence* 5(2):179-190.

Bahl LR, Brown PF, de Souza PV, Mercer RL, (1986). Maximum mutual information estimation of hidden Markov model parameters for speech recognition. In: Proc of the Int Conf on Acoustics, Speech, and Signal Processing (ICASSP'86):49-52.

Bourlard, H., Morgan, N., (1993). Continuous speech recognition by connectionist statistical methods, *IEEE Trans. on Neural Networks*, vol. 4, no. 6, pp. 893-909, 1993.

Chen, M.-Y., Kundu, A., Zhou, J., (1994). Off-line handwritten word recognition using a hidden Markov model type stochastic network, *IEEE Trans. on Pattern Analysis and Machine Intelligence*, vol. 16, no. 5:481-496.

Gilloux, M., Leroux, M., Bertille, J.-M., (1993). Strategies for handwritten words recognition using hidden Markov models, Proc. of the 2nd Int. Conf. on Document Analysis and Recognition:299-304.

Gilloux, M., Leroux, M., Bertille, J.-M., (1995a). "Strategies for Cursive Script Recognition Using Hidden Markov Models", *Machine Vision & Applications*, Special issue on Handwriting recognition, R. Plamondon ed., accepted for publication.

Gilloux, M., Lemarié, B., Leroux, M., (1995b). "A Hybrid Radial Basis Function Network/ Hidden Markov Model Handwritten Word Recognition System", Proc. of the 3rd Int. Conf. on Document Analysis and Recognition:394-397.

Gluksman, H.A., (1967). Classification of mixed font alphabetics by characteristic loci, 1st Annual IEEE Computer Conf.: 138-141.

Lemarié, B., (1993). Practical implementation of a radial basis function network for handwritten digit recognition, Proc. of the 2nd Int. Conf. on Document Analysis and Recognition:412-415.

Poggio, T., Girosi, F., (1990). Networks for approximation and learning, Proc. of the IEEE, vol 78, no 9.

Richard, M.D., Lippmann, R.P., (1991). "Neural network classifiers estimate bayesian a posteriori probabilities", *Neural Computation*, 3:461-483.

Singer, E, Lippmann, R.P., (1992). A speech recognizer using radial basis function networks in an HMM framework, Proc. of the Int. Conf. on acoustics, Speech, and Signal Processing.
